# Finite-Time Analysis of Stratified Sampling for Monte Carlo

**Alexandra Carpentier**
INRIA Lille - Nord Europe
alexandra.carpentier@inria.fr

**Rémi Munos**
INRIA Lille - Nord Europe
remi.munos@inria.fr

## Abstract

We consider the problem of stratified sampling for Monte-Carlo integration. We model this problem in a multi-armed bandit setting, where the arms represent the strata, and the goal is to estimate a weighted average of the mean values of the arms. We propose a strategy that samples the arms according to an upper bound on their standard deviations and compare its estimation quality to an ideal allocation that would know the standard deviations of the strata. We provide two regret analyses: a distribution-dependent bound $\widetilde{O}(n^{-3/2})$ that depends on a measure of the disparity of the strata, and a distribution-free bound $\widetilde{O}(n^{-4/3})$ that does not.

## 1 Introduction

Consider a polling institute that has to estimate as accurately as possible the average income of a country, given a finite budget for polls. The institute has call centers in every region in the country, and gives a part of the total sampling budget to each center so that they can call random people in the area and ask about their income. A naive method would allocate a budget proportionally to the number of people in each area. However some regions show a high variability in the income of their inhabitants whereas others are very homogeneous. Now if the polling institute knew the level of variability within each region, it could adjust the budget allocated to each region in a more clever way (allocating more polls to regions with high variability) in order to reduce the final estimation error.

This example is just one of many for which an efficient method of sampling a function with natural strata (i.e., the regions) is of great interest. Note that even in the case that there are no natural strata, it is always a good strategy to design arbitrary strata and allocate a budget to each stratum that is proportional to the size of the stratum, compared to a crude Monte-Carlo. There are many good surveys on the topic of stratified sampling for Monte-Carlo, such as (Rubinstein and Kroese, 2008)[Subsection 5.5] or (Glasserman, 2004).

The main problem for performing an efficient sampling is that the variances within the strata (in the previous example, the income variability per region) are usually unknown. One possibility is to estimate the variances *online* while sampling the strata. There is some interesting research along this direction, such as (Arouna, 2004) and more recently (Etoré and Jourdain, 2010, Kawai, 2010). The work of Etoré and Jourdain (2010) matches exactly our problem of designing an efficient adaptive sampling strategy. In this article they propose to sample according to an empirical estimate of the variance of the strata, whereas Kawai (2010) addresses a computational complexity problem which is slightly different from ours. The recent work of Etoré et al. (2011) describes a strategy that enables to sample *asymptotically* according to the (unknown) standard deviations of the strata and at the same time adapts the shape (and number) of the strata online. This is a very difficult problem, especially in high dimension, that we will not address here, although we think this is a very interesting and promising direction for further researches.

These works provide asymptotic convergence of the variance of the estimate to the targeted stratified variance[1] divided by the sample size. They also prove that the number of pulls within each stratum converges to the desired number of pulls i.e. the optimal allocation if the variances per stratum were known. Like Etoré and Jourdain (2010), we consider a stratified Monte-Carlo setting with fixed strata. Our contribution is to design a sampling strategy for which we can derive a finite-time analysis (where 'time' refers to the number of samples). This enables us to predict the quality of our estimate for any given budget $n$.

We model this problem using the setting of multi-armed bandits where our goal is to estimate a weighted average of the mean values of the arms. Although our goal is different from a usual bandit problem where the objective is to play the best arm as often as possible, this problem also exhibits an *exploration-exploitation trade-off*. The arms have to be pulled both in order to estimate the initially unknown variability of the arms (exploration) and to allocate correctly the budget according to our current knowledge of the variability (exploitation).

Our setting is close to the one described in (Antos et al., 2010) which aims at estimating *uniformly well* the mean values of all the arms. The authors present an algorithm, called GAFS-MAX, that allocates samples proportionally to the empirical variance of the arms, while imposing that each arm is pulled at least $\sqrt{n}$ times to guarantee a sufficiently good estimation of the true variances.

Note though that in the Master Thesis (Grover, 2009), the author presents an algorithm named GAFS-WL which is similar to GAFS-MAX and has an analysis close to the one of GAFS-MAX. It deals with stratified sampling, i.e. it targets an allocation which is proportional to the standard deviation (and not to the variance) of the strata times their size[2]. Some questions remain open in this work, notably that no distribution independent regret bound is provided for GAFS-WL. We clarify this point in Section 4. Our objective is similar, and we extend the analysis of this setting.

**Contributions:** In this paper, we introduce a new algorithm based on Upper-Confidence-Bounds (UCB) on the standard deviation. They are computed from the empirical standard deviation and a confidence interval derived from Bernstein's inequalities. We provide a finite-time analysis of its performance. The algorithm, called MC-UCB, samples the arms proportionally to an UCB[3] on the standard deviation times the size of the stratum. Note that the idea is similar to the one in (Carpentier et al., 2011). Our contributions are the following:

- We derive a *finite-time analysis* for the stratified sampling for Monte-Carlo setting by using an algorithm based on upper confidence bounds. We show how such a family of algorithm is particularly interesting in this setting.

- We provide two regret analysis: (i) a distribution-dependent bound $\widetilde{O}(n^{-3/2})$[4] that depends on the disparity of the stratas (a measure of the problem complexity), and which corresponds to a stationary regime where the budget $n$ is large compared to this complexity. (ii) A distribution-free bound $\widetilde{O}(n^{-4/3})$ that does not depend on the the disparity of the stratas, and corresponds to a transitory regime where $n$ is small compared to the complexity. The characterization of those two regimes and the fact that the corresponding excess error rates differ enlightens the fact that a finite-time analysis is very relevant for this problem.

The rest of the paper is organized as follows. In Section 2 we formalize the problem and introduce the notations used throughout the paper. Section 3 introduces the MC-UCB algorithm and reports performance bounds. We then discuss in Section 4 about the parameters of the algorithm and its performances. In Section 5 we report numerical experiments that

illustrate our method on the problem of pricing Asian options as introduced in (Glasserman et al., 1999). Finally, Section 6 concludes the paper and suggests future works.

## 2  Preliminaries

The allocation problem mentioned in the previous section is formalized as a $K$-armed bandit problem where each arm (stratum) $k = 1, \ldots, K$ is characterized by a distribution $\nu_k$ with mean value $\mu_k$ and variance $\sigma_k^2$. At each round $t \geq 1$, an allocation strategy (or algorithm) $\mathcal{A}$ selects an arm $k_t$ and receives a sample drawn from $\nu_{k_t}$ independently of the past samples. Note that a strategy may be adaptive, i.e., the arm selected at round $t$ may depend on past observed samples. Let $\{w_k\}_{k=1,\ldots,K}$ denote a known set of positive weights which sum to 1. For example in the setting of stratified sampling for Monte-Carlo, this would be the probability mass in each stratum. The goal is to define a strategy that estimates as precisely as possible $\mu = \sum_{k=1}^{K} w_k \mu_k$ using a total budget of $n$ samples.

Let us write $T_{k,t} = \sum_{s=1}^{t} \mathbb{I}\{k_s = k\}$ the number of times arm $k$ has been pulled up to time $t$, and $\hat{\mu}_{k,t} = \frac{1}{T_{k,t}} \sum_{s=1}^{T_{k,t}} X_{k,s}$ the empirical estimate of the mean $\mu_k$ at time $t$, where $X_{k,s}$ denotes the sample received when pulling arm $k$ for the $s$-th time.

After $n$ rounds, the algorithm $\mathcal{A}$ returns the empirical estimate $\hat{\mu}_{k,n}$ of all the arms. Note that in the case of a deterministic strategy, the expected quadratic estimation error of the weighted mean $\mu$ as estimated by the weighted average $\hat{\mu}_n = \sum_{k=1}^{K} w_k \hat{\mu}_{k,n}$ satisfies:

$$\mathbb{E}\left[\left(\hat{\mu}_n - \mu\right)^2\right] = \mathbb{E}\left[\left(\sum_{k=1}^{K} w_k(\hat{\mu}_{k,n} - \mu_k)\right)^2\right] = \sum_{k=1}^{K} w_k^2 \mathbb{E}_{\nu_k}\left[\left(\hat{\mu}_{k,n} - \mu_k\right)^2\right].$$

We thus use the following measure for the performance of any algorithm $\mathcal{A}$:

$$L_n(\mathcal{A}) = \sum_{k=1}^{K} w_k^2 \mathbb{E}\left[\left(\mu_k - \hat{\mu}_{k,n}\right)^2\right]. \tag{1}$$

The goal is to define an allocation strategy that minimizes the global loss defined in Equation 1. If the variance of the arms were known in advance, one could design an optimal static[5] allocation strategy $\mathcal{A}^*$ by pulling each arm $k$ proportionally to the quantity $w_k \sigma_k$. Indeed, if arm $k$ is pulled a deterministic number of times $T_{k,n}^*$, then

$$L_n(\mathcal{A}^*) = \sum_{k=1}^{K} w_k^2 \frac{\sigma_k^2}{T_{k,n}^*}. \tag{2}$$

By choosing $T_{k,n}^*$ such as to minimize $L_n$ under the constraint that $\sum_{k=1}^{K} T_{k,n}^* = n$, the optimal static allocation (up to rounding effects) of algorithm $\mathcal{A}^*$ is to pull each arm $k$,

$$T_{k,n}^* = \frac{w_k \sigma_k}{\sum_{i=1}^{K} w_i \sigma_i} n, \tag{3}$$

times, and achieves a global performance

$$L_n(\mathcal{A}^*) = \frac{\Sigma_w^2}{n}, \tag{4}$$

where $\Sigma_w = \sum_{i=1}^{K} w_i \sigma_i$. In the following, we write $\lambda_k = \frac{T_{k,n}^*}{n} = \frac{w_k \sigma_k}{\Sigma_w}$ the optimal allocation proportion for arm $k$ and $\lambda_{\min} = \min_{1 \leq k \leq K} \lambda_k$. Note that a small $\lambda_{\min}$ means a large disparity of the $w_k \sigma_k$ and, as explained later, provides for the algorithm we build in Section 3 a characterization of the hardness of a problem.

However, in the setting considered here, the $\sigma_k$ are unknown, and thus the optimal allocation is out of reach. A possible allocation is the uniform strategy $\mathcal{A}^u$, i.e., such that $T_k^u = \frac{w_k}{\sum_{i=1}^{K} w_i} n$. Its performance is

$$L_n(\mathcal{A}^u) = \sum_{k=1}^{K} w_k \sum_{k=1}^{K} \frac{w_k \sigma_k^2}{n} = \frac{\Sigma_{w,2}}{n},$$

where $\Sigma_{w,2} = \sum_{k=1}^{K} w_k \sigma_k^2$. Note that by Cauchy-Schwartz's inequality, we have $\Sigma_w^2 \leq \Sigma_{w,2}$ with equality if and only if the $(\sigma_k)$ are all equal. Thus $\mathcal{A}^*$ is always at least as good as $\mathcal{A}^u$. In addition, since $\sum_i w_i = 1$, we have $\Sigma_w^2 - \Sigma_{w,2} = -\sum_k w_k(\sigma_k - \Sigma_w)^2$. The difference between those two quantities is the weighted quadratic variation of the $\sigma_k$ around their weighted mean $\Sigma_w$. In other words, it is the variance of the $(\sigma_k)_{1 \leq k \leq K}$. As a result the gain of $\mathcal{A}^*$ compared to $\mathcal{A}^u$ grow with the disparity of the $\sigma_k$.

We would like to do better than the uniform strategy by considering an adaptive strategy $\mathcal{A}$ that would estimate the $\sigma_k$ at the same time as it tries to implement an allocation strategy as close as possible to the optimal allocation algorithm $\mathcal{A}^*$. This introduces a natural trade-off between the exploration needed to improve the estimates of the variances and the exploitation of the current estimates to allocate the pulls nearly-optimally.

In order to assess how well $\mathcal{A}$ solves this trade-off and manages to sample according to the true standard deviations *without knowing them in advance*, we compare its performance to that of the optimal allocation strategy $\mathcal{A}^*$. For this purpose we define the notion of *regret* of an adaptive algorithm $\mathcal{A}$ as the difference between the performance loss incurred by the algorithm and the optimal algorithm:

$$R_n(\mathcal{A}) = L_n(\mathcal{A}) - L_n(\mathcal{A}^*). \tag{5}$$

The *regret* indicates how much we loose in terms of expected quadratic estimation error by not knowing in advance the standard deviations $(\sigma_k)$. Note that since $L_n(\mathcal{A}^*) = \frac{\Sigma_w^2}{n}$, a consistent strategy i.e., asymptotically equivalent to the optimal strategy, is obtained whenever its regret is neglectable compared to $1/n$.

## 3 Allocation based on Monte Carlo Upper Confidence Bound

### 3.1 The algorithm

In this section, we introduce our adaptive algorithm for the allocation problem, called *Monte Carlo Upper Confidence Bound* (MC-UCB). The algorithm computes a high-probability bound on the standard deviation of each arm and samples the arms proportionally to their bounds times the corresponding weights. The MC-UCB algorithm, $\mathcal{A}_{MC-UCB}$, is described in Figure 1. It requires three parameters as inputs: $c_1$ and $c_2$ which are related to the shape of the distributions (see Assumption 1), and $\delta$ which defines the *confidence level* of the bound. In Subsection 4.2, we discuss a way to reduce the number of parameters from three to one. The amount of exploration of the algorithm can be adapted by properly tuning these parameters.

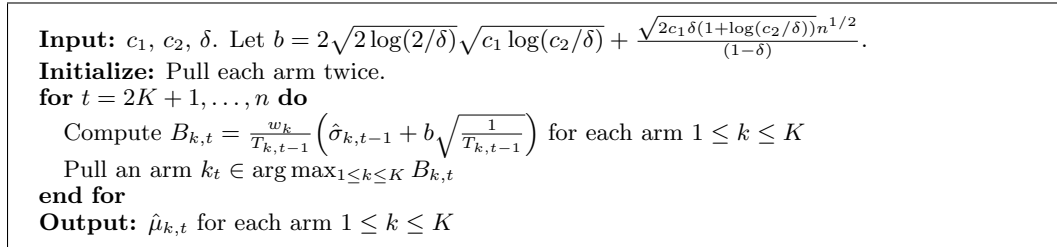

Figure 1: The pseudo-code of the MC-UCB algorithm. The empirical standard deviations $\hat{\sigma}_{k,t-1}$ are computed using Equation 6.

The algorithm starts by pulling each arm twice in rounds $t = 1$ to $2K$. From round $t = 2K+1$ on, it computes an upper confidence bound $B_{k,t}$ on the standard deviation $\sigma_k$, for each arm $k$, and then pulls the one with largest $B_{k,t}$. The upper bounds on the standard deviations are built by using Theorem 10 in (Maurer and Pontil, 2009)[6] and based on the empirical standard deviation $\hat{\sigma}_{k,t-1}$ :

$$\hat{\sigma}_{k,t-1}^2 = \frac{1}{T_{k,t-1} - 1} \sum_{i=1}^{T_{k,t-1}} (X_{k,i} - \hat{\mu}_{k,t-1})^2, \tag{6}$$

where $X_{k,i}$ is the $i$-th sample received when pulling arm $k$, and $T_{k,t-1}$ is the number of pulls allocated to arm $k$ up to time $t-1$. After $n$ rounds, MC-UCB returns the empirical mean $\hat{\mu}_{k,n}$ for each arm $1 \leq k \leq K$.

## 3.2 Regret analysis of MC-UCB

Before stating the main results of this section, we state the assumption that the distributions are sub-Gaussian, which includes e.g., Gaussian or bounded distributions. See (Buldygin and Kozachenko, 1980) for more precisions.

**Assumption 1** *There exist $c_1, c_2 > 0$ such that for all $1 \leq k \leq K$ and any $\epsilon > 0$,*

$$\mathbb{P}_{X \sim \nu_k}(|X - \mu_k| \geq \epsilon) \leq c_2 \exp(-\epsilon^2/c_1) . \tag{7}$$

We provide two analyses, a *distribution-dependent* and a *distribution-free*, of MC-UCB, which are respectively interesting in two *regimes*, i.e., stationary and transitory *regimes*, of the algorithm. We will comment on this later in Section 4.

**A *distribution-dependent* result:** We now report the first bound on the regret of MC-UCB algorithm. The proof is reported in (Carpentier and Munos, 2011). and relies on upper- and lower-bounds on $T_{k,t} - T_{k,t}^*$, i.e., the difference in the number of pulls of each arm compared to the optimal allocation (see Lemma 3).

**Theorem 1** *Under Assumption 1 and if we choose $c_2$ such that $c_2 \geq 2Kn^{-5/2}$, the regret of MC-UCB run with parameter $\delta = n^{-7/2}$ with $n \geq 4K$ is bounded as*

$$R_n(\mathcal{A}_{MC-UCB}) \leq \frac{\log(n)c_1(c_2+2)}{n^{3/2}\lambda_{\min}^{3/2}}\Big(112\Sigma_w + 6K\Big) + \frac{19}{\lambda_{\min}^3 n^2}\Big(K\Sigma_w^2 + 720c_1(c_2+1)\log(n)^2\Big).$$

Note that this result crucially depends on the smallest proportion $\lambda_{\min}$ which is a measure of the disparity of the standard deviations times their weight. For this reason we refer to it as "distribution-dependent" result.

**A *distribution-free* result:** Now we report our second regret bound that does not depend on $\lambda_{\min}$ but whose rate is poorer. The proof is reported in (Carpentier and Munos, 2011) and relies on other upper- and lower-bounds on $T_{k,t} - T_{k,t}^*$ detailed in Lemma 4.

**Theorem 2** *Under Assumption 1 and if we choose $c_2$ such that $c_2 \geq 2Kn^{-5/2}$, the regret of MC-UCB run with parameter $\delta = n^{-7/2}$ with $n \geq 4K$ is bounded as*

$$R_n(\mathcal{A}_{MC-UCB}) \leq \frac{200\sqrt{c_1}(c_2+2)\Sigma_w K}{n^{4/3}}\log(n) + \frac{365}{n^{3/2}}\Big(129c_1(c_2+2)^2K^2\log(n)^2 + K\Sigma_w^2\Big).$$

This bound does not depend on $1/\lambda_{\min}$. Note that the bound is not entirely distribution free since $\Sigma_w$ appears. But it can be proved using Assumption 1 that $\Sigma_w^2 \leq c_1 c_2$. This is obtained at the price of the slightly worse rate $\widetilde{O}(n^{-4/3})$.

# 4 Discussion on the results

## 4.1 Distribution-free versus distribution-dependent

Theorem 1 provides a regret bound of order $\widetilde{O}(\lambda_{\min}^{-5/2}n^{-3/2})$, whereas Theorem 2 provides a bound in $\widetilde{O}(n^{-4/3})$ independently of $\lambda_{\min}$. Hence, for a given problem i.e., a given $\lambda_{\min}$, the distribution-free result of Theorem 2 is more informative than the distribution-dependent result of Theorem 1 in the *transitory regime*, that is to say when $n$ is small compared to $\lambda_{\min}^{-1}$. The distribution-dependent result of Theorem 1 is better in the *stationary regime* i.e., for large $n$. This distinction reminds us of the difference between distribution-dependent and distribution-free bounds for the UCB algorithm in usual multi-armed bandits[7].

Although we do not have a lower bound on the regret yet, we believe that the rate $n^{-3/2}$ cannot be improved for general distributions. As explained in the proof in Appendix B of (Carpentier and Munos, 2011), this rate is a direct consequence of the high probability bounds on the estimates of the standard deviations of the arms which are in $O(1/\sqrt{n})$, *and those bounds are tight.* A natural question is whether there exists an algorithm with a regret of order $\widetilde{O}(n^{-3/2})$ without any dependence in $\lambda_{\min}^{-1}$. Although we do not have an answer to this question, we can say that our algorithm MC-UCB does not satisfy this property. In Appendix D.1 of (Carpentier and Munos, 2011), we give a simple example where $\lambda_{\min} = 0$ and for which the rate of MC-UCB cannot be better than $\widetilde{O}(n^{-4/3})$. This shows that our analysis of MC-UCB is tight.

The problem dependent upper bound is similar to the one provided for GAFS-WL in (Grover, 2009). We however expect that GAFS-WL has for some problems a sub-optimal behavior: it is possible to find cases where $R_n(\mathcal{A}_{GAFS-WL}) = \Omega(1/n)$, see Appendix D.1 of (Carpentier and Munos, 2011). Note however that when there is an arm with 0 standard deviation, GAFS-WL is likely to perform better than MC-UCB, as it will only sample this arm $O(\sqrt{n})$ times while MC-UCB samples it $\widetilde{O}(n^{2/3})$ times.

## 4.2 The parameters of the algorithm

Our algorithm takes three parameters as input, namely $c_1$, $c_2$ and $\delta$, but we only use a combination of them in the algorithm, with the introduction of $b = 2\sqrt{2\log(2/\delta)}\sqrt{c_1\log(c_2/\delta)} + \frac{\sqrt{2c_1\delta(1+\log(c_2/\delta))}n^{1/2}}{(1-\delta)}$. For practical use of the method, it is enough to tune the algorithm with a single parameter $b$. By the choice of the value assigned to $\delta$ in the two theorems, $b$ should be chosen of order $c\log(n)$, where $c$ can be interpreted as a high probability bound on the range of the samples. We thus simply require a rough estimate of the magnitude of the samples. Note that in the case of bounded distributions, $b$ can be chosen as $b = 4\sqrt{\frac{5}{2}}c\sqrt{\log(n)}$ where $c$ is a true bound on the variables. This result is easy to deduce by simplifying Lemma 1 in Appendix A of (Carpentier and Munos, 2011) for the case of bounded variables.

## 5 Numerical experiment: Pricing of an Asian option

We consider the pricing problem of an Asian option introduced in (Glasserman et al., 1999) and later considered in (Kawai, 2010, Etoré and Jourdain, 2010). This uses a Black-Schole model with strike $C$ and maturity $T$. Let $(W(t))_{0\leq t\leq 1}$ be a Brownian motion that is discretized at $d$ equidistant times $\{i/d\}_{1\leq i\leq d}$, which defines the vector $W \in \mathbb{R}^d$ with components $W_i = W(i/d)$. The discounted payoff of the Asian option is defined as a function of $W$, by:

$$F(W) = \exp(-rT)\max\left[\frac{1}{d}\sum_{i=1}^d S_0 \exp\left[(r - \frac{1}{2}s_0^2)\frac{iT}{d} + s_0\sqrt{T}W_i\right] - C, 0\right], \quad (8)$$

where $S_0$, $r$, and $s_0$ are constants, and the price is defined by the expectation $p = \mathbb{E}_W F(W)$.

We want to estimate the price $p$ by Monte-Carlo simulations (by sampling on $W = (W_i)_{1\leq i\leq d}$). In order to reduce the variance of the estimated price, we can stratify the space of $W$. Glasserman et al. (1999) suggest to stratify according to a one dimensional projection of $W$, i.e., by choosing a projection vector $u \in \mathbb{R}^d$ and define the strata as the set of $W$ such that $u \cdot W$ lies in intervals of $\mathbb{R}$. They further argue that the best direction for stratification is to choose $u = (0, \cdots, 0, 1)$, i.e., to stratify according to the last component $W_d$ of $W$. Thus we sample $W_d$ and then conditionally sample $W_1, ..., W_{d-1}$ according to a Brownian Bridge as explained in (Kawai, 2010). Note that this choice of stratification is also intuitive since $W_d$ has the largest exponent in the payoff (8), and thus the highest volatility. Kawai (2010) and Etoré and Jourdain (2010) also use the same direction of stratification.

Like in (Kawai, 2010) we consider 5 strata of equal weight. Since $W_d$ follows a $\mathcal{N}(0,1)$, the strata correspond to the 20-percentile of a normal distribution. The left plot of Figure 2 represents the cumulative distribution function of $W_d$ and shows the strata in terms of

percentiles of $W_d$. The right plot represents, in dot line, the curve $\mathbb{E}[F(W)|W_d = x]$ versus $\mathbb{P}(W_d < x)$ parameterized by $x$, and the box plot represents the expectation and standard deviations of $F(W)$ conditioned on each stratum. We observe that this stratification produces an important heterogeneity of the standard deviations per stratum, which indicates that a stratified sampling would be profitable compared to a crude Monte-Carlo sampling.

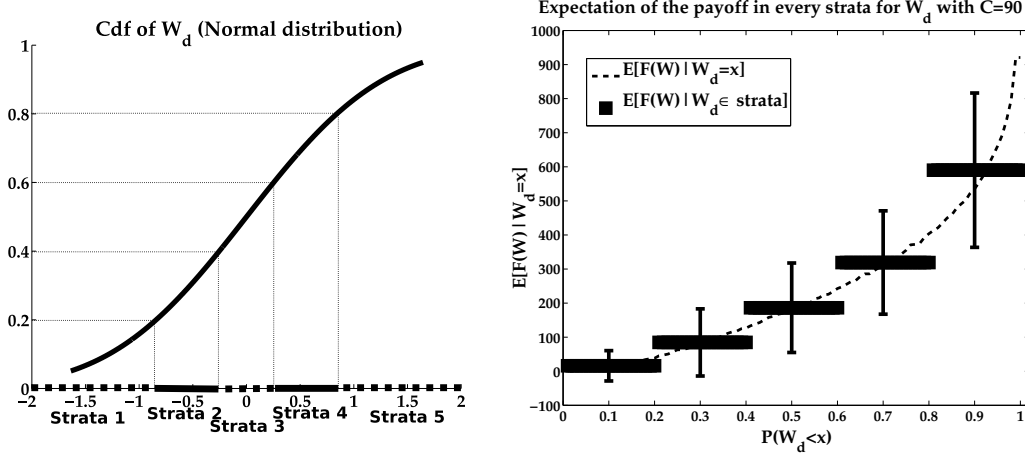

Figure 2: Left: Cdf of $W_d$ and the definition of the strata. Right: expectation and standard deviation of $F(W)$ conditioned on each stratum for a strike $C = 90$.

We choose the same numerical values as Kawai (2010): $S_0 = 100$, $r = 0.05$, $s_0 = 0.30$, $T = 1$ and $d = 16$. Note that the strike $C$ of the option has a direct impact on the variability of the strata. Indeed, the larger $C$, the more probable $F(W) = 0$ for strata with small $W_d$, and thus, the smaller $\lambda_{\min}$.

Our two main competitors are the SSAA algorithm of Etoré and Jourdain (2010) and GAFS-WL of Grover (2009). We did not compare to (Kawai, 2010) which aims at minimizing the computational time and not the loss considered here[8]. SSAA works in $K_r$ rounds of length $N_k$ where, at each round, it allocates proportionally to the empirical standard deviations computed in the previous rounds. Etoré and Jourdain (2010) report the asymptotic consistency of the algorithm whenever $\frac{k}{N_k}$ goes to 0 when $k$ goes to infinity. Since their goal is not to obtain a finite-time performance, they do not mention how to calibrate the length and number of rounds in practice. We choose the same parameters as in their numerical experiments (Section 3.2.2 of (Etoré and Jourdain, 2010)) using 3 rounds. In this setting where we know the budget $n$ at the beginning of the algorithm, GAFS-WL pulls each arm $a\sqrt{n}$ times and then pulls at time $t + 1$ the arm $k_{t+1}$ that maximizes $\frac{w_k \hat{\sigma}_{k,t}}{T_{k,t}}$. We set $a = 1$.

As mentioned in Subsection 4.2, an advantage of our algorithm is that it requires a single parameter to tune. We chose $b = 1000 \log(n)$ where 1000 is a high-probability range of the variables (see right plot of Figure 2). Table 5 reports the performance of MC-UCB, GAFS-WL, SSAA, and the uniform strategy, for different values of strike $C$ i.e., for different values of $\lambda_{\min}^{-1}$ and $\Sigma_{w,2}/\Sigma_w^2 = \frac{\sum w_k \sigma_k^2}{(\sum_k w_k \sigma_k)^2}$. The total budget is $n = 10^5$. The results are averaged on 50000 trials. We notice that MC-UCB outperforms SSAA, the uniform strategy, and GAFS-WL strategy. Note however that, in the case of GAFS-WL strategy, the small gain could come from the fact that there are more parameters in MC-UCB, and that we were thus able to adjust them (even if we kept the same parameters for the three values of C).

In the left plot of Figure 3, we plot the rescaled regret $R_n n^{3/2}$, averaged over 50000 trials, as a function of $n$, where $n$ ranges from 50 to 5000. The value of the strike is $C = 120$. Again, we notice that MC-UCB performs better than Uniform and SSAA because it adapts

| $C$ | $\frac{1}{\lambda_{\min}}$ | $\Sigma_{w,2}/\Sigma_w^2$ | Uniform | SSAA | GAFS-WL | MC-UCB |
|---|---|---|---|---|---|---|
| 60 | 6.18 | 1.06 | $2.52\ 10^{-2}$ | $5.87\ 10^{-3}$ | $8.25\ 10^{-4}$ | $7.29\ 10^{-4}$ |
| 90 | 15.29 | 1.24 | $3.32\ 10^{-2}$ | $6.14\ 10^{-3}$ | $8.58\ 10^{-4}$ | $8.07\ 10^{-4}$ |
| 120 | 744.25 | 3.07 | $3.56\ 10^{-2}$ | $6.22\ 10^{-3}$ | $9.89\ 10^{-4}$ | $9.28\ 10^{-4}$ |

Table 1: Characteristics of the distributions ($\lambda_{\min}^{-1}$ and $\Sigma_{w,2}/\Sigma_w^2$) and regret of the Uniform, SSAA, and MC-UCB strategies, for different values of the strike $C$.

faster to the distributions of the strata. But it performs very similarly to GAFS-WL. In addition, it seems that the regret of Uniform and SSAA grows faster than the rate $n^{3/2}$, whereas MC-UCB, as well as GAFS-WL, grow with this rate. The right plot focuses on the MC-UCB algorithm and rescales the $y-$axis to observe the variations of its rescaled regret more accurately. The curve grows first and then stabilizes. This could correspond to the two regimes discussed previously.

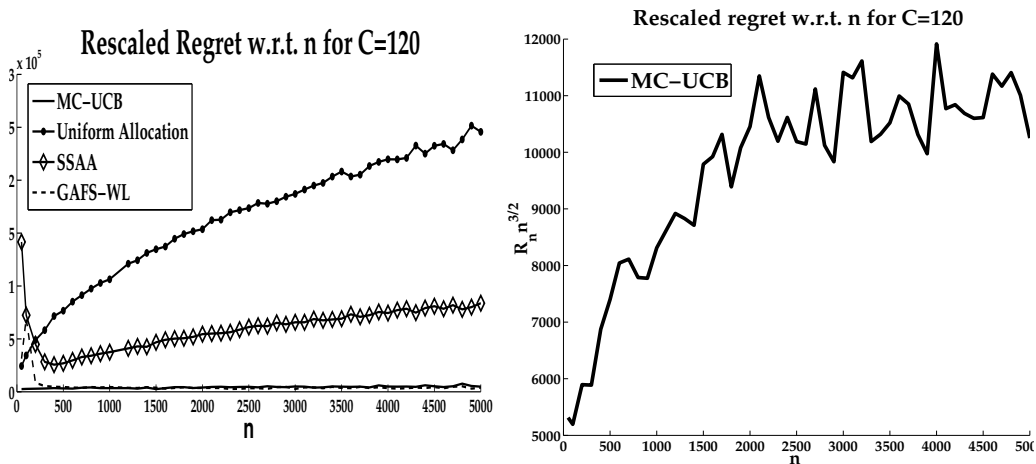

Figure 3: Left: Rescaled regret ($R_n n^{3/2}$) of the Uniform, SSAA, and MC-UCB strategies. Right: zoom on the rescaled regret for MC-UCB that illustrates the two regimes.

## 6   Conclusions

We provided a finite-time analysis for stratified sampling for Monte-Carlo in the case of fixed strata. We reported two bounds: (i) a distribution dependent bound $\widetilde{O}(n^{-3/2}\lambda_{\min}^{-5/2})$ which is of interest when $n$ is large compared to a measure of disparity $\lambda_{\min}^{-1}$ of the standard deviations (*stationary regime*), and (ii) a distribution free bound in $\widetilde{O}(n^{-4/3})$ which is of interest when $n$ is small compared to $\lambda_{\min}^{-1}$ (*transitory regime*).

Possible directions for future work include: (i) making the MC-UCB algorithm anytime (i.e. not requiring the knowledge of $n$), (ii) investigating whether their exists an algorithm with $\widetilde{O}(n^{-3/2})$ regret without dependency on $\lambda_{\min}^{-1}$, and (iii) deriving distribution-dependent and distribution-free lower-bounds for this problem.

**Acknowledgements**

We thank András Antos for several comments that helped us to improve the quality of the paper. This research was partially supported by Region Nord-Pas-de-Calais Regional Council, French ANR EXPLO-RA (ANR-08-COSI-004), the European Communitys Seventh Framework Programme (FP7/2007-2013) under grant agreement 231495 (project CompLACS), and by Pascal-2.

## Footnotes

[1]The target is defined in [Subsection 5.5] of (Rubinstein and Kroese, 2008) and later in this paper, see Equation 4.

[2]This is explained in (Rubinstein and Kroese, 2008) and will be formulated precisely later.

[3]Note that we consider a sampling strategy based on UCBs on the standard deviations of the arms whereas the so-called *UCB algorithm* of Auer et al. (2002), in the usual multi-armed bandit setting, computes UCBs on the mean rewards of the arms.

[4]The notation $\widetilde{O}(\cdot)$ corresponds to $O(\cdot)$ up to logarithmic factors.

[5]Static means that the number of pulls allocated to each arm does not depend on the received samples.

[6]We could also have used the variant reported in (Audibert et al., 2009).

[7]The distribution dependent bound is in $O(K \log n/\Delta)$, where $\Delta$ is the difference between the mean value of the two best arms, and the distribution-free bound is in $O(\sqrt{nK \log n})$ as explained in (Auer et al., 2002, Audibert and Bubeck, 2009).

[8]In that paper, the computational costs for each stratum vary, i.e. it is faster to sample in some strata than in others, and the aim of their paper is to minimize the global computational cost while achieving a given performance.

# References

András Antos, Varun Grover, and Csaba Szepesvári. Active learning in heteroscedastic noise. *Theoretical Computer Science*, 411:2712–2728, June 2010.

B. Arouna. Adaptative monte carlo method, a variance reduction technique. *Monte Carlo Methods and Applications*, 10(1):1–24, 2004.

J.Y. Audibert and S. Bubeck. Minimax policies for adversarial and stochastic bandits. In *22nd annual conference on learning theory*, 2009.

J.Y. Audibert, R. Munos, and Cs. Szepesvári. Exploration-exploitation tradeoff using variance estimates in multi-armed bandits. *Theoretical Computer Science*, 410(19):1876–1902, 2009.

P. Auer, N. Cesa-Bianchi, and P. Fischer. Finite-time analysis of the multiarmed bandit problem. *Machine learning*, 47(2):235–256, 2002.

VV Buldygin and Y.V. Kozachenko. Sub-gaussian random variables. *Ukrainian Mathematical Journal*, 32(6):483–489, 1980.

A. Carpentier and R. Munos. Finite-time analysis of stratified sampling for monte carlo. Technical Report inria-00636924, INRIA, 2011.

A. Carpentier, A. Lazaric, M. Ghavamzadeh, R. Munos, and P. Auer. Upper-confidence-bound algorithms for active learning in multi-armed bandits. In *Algorithmic Learning Theory*, pages 189–203. Springer, 2011.

Pierre Etoré and Benjamin Jourdain. Adaptive optimal allocation in stratified sampling methods. *Methodol. Comput. Appl. Probab.*, 12(3):335–360, September 2010.

Pierre Etoré, Gersende Fort, Benjamin Jourdain, and Éric Moulines. On adaptive stratification. *Ann. Oper. Res.*, 2011. to appear.

P. Glasserman. *Monte Carlo methods in financial engineering*. Springer Verlag, 2004. ISBN 0387004513.

P. Glasserman, P. Heidelberger, and P. Shahabuddin. Asymptotically optimal importance sampling and stratification for pricing path-dependent options. *Mathematical Finance*, 9 (2):117–152, 1999.

V. Grover. Active learning and its application to heteroscedastic problems. *Department of Computing Science, Univ. of Alberta, MSc thesis*, 2009.

R. Kawai. Asymptotically optimal allocation of stratified sampling with adaptive variance reduction by strata. *ACM Transactions on Modeling and Computer Simulation (TOMACS)*, 20(2):1–17, 2010. ISSN 1049-3301.

A. Maurer and M. Pontil. Empirical bernstein bounds and sample-variance penalization. In *Proceedings of the Twenty-Second Annual Conference on Learning Theory*, pages 115–124, 2009.

S.I. Resnick. *A probability path*. Birkhäuser, 1999.

R.Y. Rubinstein and D.P. Kroese. *Simulation and the Monte Carlo method*. Wiley-interscience, 2008. ISBN 0470177942.

